# Distributed Probabilistic Learning for Camera Networks with Missing Data

**Sejong Yoon**
Department of Computer Science
Rutgers University
sjyoon@cs.rutgers.edu

**Vladimir Pavlovic**
Department of Computer Science
Rutgers University
vladimir@cs.rutgers.edu

## Abstract

Probabilistic approaches to computer vision typically assume a centralized setting, with the algorithm granted access to all observed data points. However, many problems in wide-area surveillance can benefit from distributed modeling, either because of physical or computational constraints. Most distributed models to date use algebraic approaches (such as distributed SVD) and as a result cannot explicitly deal with missing data. In this work we present an approach to estimation and learning of generative probabilistic models in a distributed context where certain sensor data can be missing. In particular, we show how traditional centralized models, such as probabilistic PCA and missing-data PPCA, can be learned when the data is distributed across a network of sensors. We demonstrate the utility of this approach on the problem of distributed affine structure from motion. Our experiments suggest that the accuracy of the learned probabilistic structure and motion models rivals that of traditional centralized factorization methods while being able to handle challenging situations such as missing or noisy observations.

## 1 Introduction

Traditional computer vision algorithms, particularly those that exploit various probabilistic and learning-based approaches, are often formulated in centralized settings. A scene or an object is observed by a single camera with all acquired information centrally processed and stored in a single knowledge base (e.g., a classification model). Even if the problem setting relies on multiple cameras, as may be the case in multi-view or structure from motion (SfM) tasks, all collected information is still processed and organized in a centralized fashion. Increasingly modern computational settings are becoming characterized by networks of peer-to-peer connected devices, with local data processing abilities. Nevertheless, the overall goal of such *distributed device (camera) networks* may still be to exchange information and form a consensus interpretation of the visual scene. For instance, even if a camera observes a limited set of object views, one would like its local computational model to reflect a general 3D appearance of the object visible by other cameras in the network.

A number of distributed algorithms have been proposed to address the problems such as calibration, pose estimation, tracking, object and activity recognition in large camera networks [1–3]. In order to deal with high dimensionality of vision problems, distributed latent space search such as decentralized variants of PCA have been studied in [4, 5]. A more general framework using distributed least squares [6] based on distributed averaging of sensor fusions [7] was introduced for PCA, triangulation, pose estimation and SfM. Similar approaches have been extended to settings such as the distributed object tracking and activity interpretation [8,9]. Even though the methods such as PCA or Kalman filtering have their well-known probabilistic counterparts, the aforementioned approaches do not use probabilistic formulation when dealing with the distributed setting.

One critical challenge in distributed data analysis includes dealing with missing data. In camera networks, different nodes will only have access to a partial set of data features because of varying camera views or object movement. For instance, object points used for SfM may be visible only

in some cameras and only in particular object poses. As a consequence, different nodes will be frequently exposed to missing data. However, most current distributed data analysis methods are algebraic in nature and cannot seamlessly handle such missing data.

In this work we propose a distributed consensus learning approach for parametric probabilistic models with latent variables that can effectively deal with missing data. We assume that each node in a network can observe only a fraction of the data (e.g., object views in camera networks). Furthermore, we assume that some of the data features may be missing across different nodes. The goal of the network of sensors is to learn a single consensus probabilistic model (e.g., 3D object structure) without ever resorting to a centralized data pooling and centralized computation. We will demonstrate that this task can be accomplished in a principled manner by local probabilistic models and in-network information sharing, implemented as recursive distributed probabilistic learning.

In particular, we focus on probabilistic PCA (PPCA) as a prototypical example and derive its distributed version, the D-PPCA. We then suggest how missing data can be handled in this setting using a missing-data PPCA and apply this model to solve the distributed SfM task in a camera network. Our model is inspired by the consensus-based distributed Expectation-Maximization (EM) algorithm for Gaussian mixtures [10], which we extend to deal with generalized linear Gaussian models [11]. Unlike other recently proposed decomposable Gaussian graphical models [4, 12], our model does not depend on any specific type of graphs. Our network, of arbitrary topology, is assumed to be static with a single connected component. These assumptions are reasonably applicable to many real world camera network settings.

In Section 2, we first explain the general distributed probabilistic model. Section 3 shows how D-PPCA can be formulated as a special case of the probabilistic framework and propose the means for handling missing data. We then explain how D-PPCA can be modified for the application in affine SfM. In Section 5, we report experimental results of our model using both synthetic and real data. Finally, we discuss our approach including its limitations and possible solutions in Section 6.

## 2   Distributed Probabilistic Model

We start our discussion by first considering a general parametric probabilistic model in a centralized setting and then we show how to derive its distributed form.

### 2.1   Centralized Setting

Let $\mathbf{X} = \{\mathbf{x}_n | \mathbf{x}_n \in \mathcal{R}^D\}$ be a set of iid multivariate data points with the corresponding latent variables $\mathbf{Z} = \{\mathbf{z}_n | \mathbf{z}_n \in \mathcal{R}^M\}$, $n = 1...N$. Our model is a joint density defined on $(\mathbf{x}_n, \mathbf{z}_n)$ with a global parameter $\theta$

$$(\mathbf{x}_n, \mathbf{z}_n) \sim p(\mathbf{x}_n, \mathbf{z}_n | \theta),$$

with $p(\mathbf{X}, \mathbf{Z} | \theta) = \prod_n p(\mathbf{x}_n, \mathbf{z}_n | \theta)$, as depicted in Fig. 1a. In this general model, we can find an optimal global parameter $\hat{\theta}$ (in a MAP sense) by applying standard EM learning. The EM follows a recursive two-step procedure: (a) E-step, where the posterior density $p(\mathbf{z}_n | \mathbf{x}_n, \theta)$ is estimated, and (b) M-step: parametric optimization $\hat{\theta} = \arg\max_\theta \mathbb{E}_{\mathbf{Z}|\mathbf{X}} [\log p(\mathbf{X}, \mathbf{Z} | \theta)]$. It is important to point out that each posterior density estimate at point $n$ depends solely on the corresponding measurement $\mathbf{x}_n$ and does not depend on any other $\mathbf{x}_k, k \neq n$. This means that even if we partition independent measurements into arbitrary subsets, posterior density estimation is accomplished locally, within each subset. However, in the M-step all measurements $\mathbf{X}$ affect the choice of $\hat{\theta}$ because of the dependence of each term in the completed log likelihood on the same $\hat{\theta}$. This is a typical characteristic of parametric models where the optimal parameters depend on summary data statistics.

### 2.2   Distributed Setting

Let $G = (V, E)$ be an undirected connected graph with vertices $i, j \in V$ and edges $e_{ij} = (i, j) \in E$ connecting the two vertices. Each $i$-th node is directly connected with 1-hop neighbors in $\mathcal{B}_i = \{j | e_{ij} \in E\}$. Suppose the set of data samples at $i$-th node is $\mathbf{X}_i = \{\mathbf{x}_{in} | n = 1, ..., N_i\}$, where $\mathbf{x}_{in} \in \mathcal{R}^D$ is $n$-th measurement vector and $N_i$ is the number of samples collected in $i$-th node. Likewise, we define the latent variable set for node $i$ as $\mathbf{Z}_i = \{\mathbf{z}_{in} | n = 1, ..., N_i\}$.

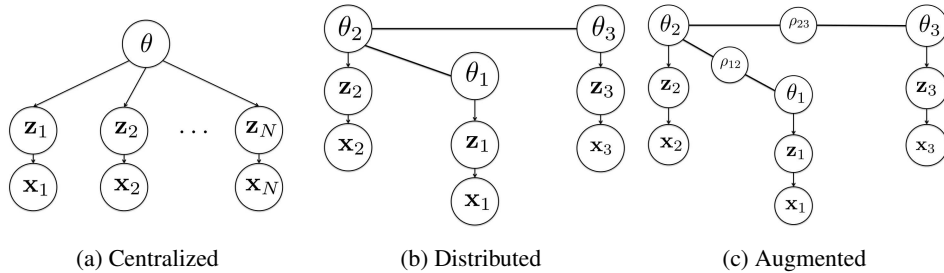

<div align="center">

(a) Centralized      (b) Distributed      (c) Augmented

Figure 1: Centralized, distributed and augmented models for probabilistic PCA.

</div>

As observed previously, each posterior estimation is decentralized. Learning the model parameter would be decentralized if each node had its own independent parameter $\theta_i$. Still, the centralized model can be equivalently defined using the set of local parameters, with an additional constraint on their consensus, $\theta_1 = \theta_2 = \cdots = \theta_{|V|}$. This is illustrated in Fig. 1b where the local node models are constrained using ties defined on the underlying graph. The simple consensus tying can be more conveniently defined using a set of auxiliary variables $\rho_{ij}$, one for each edge $e_{ij}$ (Fig. 1c). This now leads to the final distributed consensus learning formulation, similar to [10]:

$$\hat{\boldsymbol{\theta}} = \arg\min_{\{\theta_i : i \in V\}} -\log p(\mathbf{X}|\boldsymbol{\theta}, G) \quad s.t. \quad \theta_i = \rho_{ij}, \rho_{ij} = \theta_j, i \in V, j \in \mathcal{B}_i \tag{1}$$

where we marginalized on $\mathbf{X}$. This is a constrained optimization task that can be solved in a principal manner using the Alternating Direction Method of Multipliers (ADMM) [13–15]. ADMM iteratively, in a block-coordinate fashion, solves $\max_\lambda \min_\theta \mathcal{L}(\cdot)$ on the augmented Lagrangian

$$\mathcal{L}(\boldsymbol{\theta}, \rho, \lambda) = -\log p(\mathbf{X}|\theta_1, \theta_2, ..., \theta_{|V|}, G) + \sum_{i \in V} \sum_{j \in \mathcal{B}_i} \left\{ \lambda_{ij1}^{\mathrm{T}}(\theta_i - \rho_{ij}) + \lambda_{ij2}^{\mathrm{T}}(\rho_{ij} - \theta_j) \right\}$$

$$+ \frac{\eta}{2} \sum_{i \in V} \sum_{j \in \mathcal{B}_i} \left\{ ||\theta_i - \rho_{ij}||^2 + ||\rho_{ij} - \theta_j||^2 \right\} \tag{2}$$

where $\lambda_{ij1}, \lambda_{ij2}, i, j \in V$ are the Lagrange multipliers, $\eta$ is some positive scalar parameter and $|| \cdot ||$ is induced norm. The last term (modulated by $\eta$) is not strictly necessary for consensus but introduces additional regularization. Further discussions on this term and the parameter can be found in [15] and [16]. The auxiliary $\rho_{ij}$ play a critical decoupling role and separate estimation of local $\theta_i$ during block-coordinate ascent/descent. This classic (first introduced in 1970s) meta decompose algorithm can be used to devise a distributed counterpart for any centralized problem that attempts to maximize a global log likehood function over a connected network.

## 3 Distributed Probabilistic PCA (D-PPCA)

We now apply the general distributed probabilistic learning explained above to the specific case of distributed PPCA. Traditional centralized formulation of probabilistic PCA (PPCA) [17] assumes that latent variable $\mathbf{z}_{in} \sim \mathcal{N}(\mathbf{z}_{in}|0, \mathbf{I})$, with a generative relation

$$\mathbf{x}_{in} = \mathbf{W}_i \mathbf{z}_{in} + \boldsymbol{\mu}_i + \boldsymbol{\epsilon}_i, \tag{3}$$

where $\boldsymbol{\epsilon}_i \sim \mathcal{N}(\boldsymbol{\epsilon}_i|0, a_i^{-1}\mathbf{I})$ and $a_i$ is the noise precision. Inference then yields

$$p(\mathbf{z}_{in}|\mathbf{x}_{in}) = \mathcal{N}(\mathbf{z}_{in}|\mathbf{L}_i^{-1}\mathbf{W}_i^{\mathrm{T}}(\mathbf{x}_{in} - \boldsymbol{\mu}_i), a_i^{-1}\mathbf{L}_i^{-1}), \tag{4}$$

where $\mathbf{L}_i = \mathbf{W}_i^{\mathrm{T}}\mathbf{W}_i + a_i^{-1}\mathbf{I}$. We can find optimal parameters $\mathbf{W}_i, \boldsymbol{\mu_i}, a_i$ by finding the maximum likelihood estimates of the marginal data likelihood or by applying the EM algorithm on expected complete data log likelihood with respect to the posterior density $p(\mathbf{Z}_i|\mathbf{X}_i)$.

### 3.1 Distributed Formulation

The distributed algorithm developed in Section 2 can be directly applied to this PPCA model. The basic idea is to assign each subset of samples as evidence for the local generative models with

<div align="center">3</div>

parameters $\mathbf{W}_i, \boldsymbol{\mu}_i, a_i^{-1}$. The inference is accomplished locally in each node. The local parameter estimates are then computed using the consensus updates that combine local summary data statistics with the information about the model conveyed through neighboring network nodes. Below, we outline specific details of this approach.

Let $\boldsymbol{\Theta}_i = \{\mathbf{W}_i, \boldsymbol{\mu}_i, a_i\}$ be the set of parameters for each node $i$. The global constrained consensus optimization now becomes

$$\min_{\{\mathbf{W}_i, \boldsymbol{\mu}_i, a_i : i \in V\}} -F(\boldsymbol{\Theta}_i) \quad s.t. \quad \begin{array}{ll} \mathbf{W}_i = \boldsymbol{\rho}_{ij}, & \boldsymbol{\rho}_{ij} = \mathbf{W}_j, \quad i \in V, j \in \mathcal{B}_i, \\ \boldsymbol{\mu}_i = \boldsymbol{\phi}_{ij}, & \boldsymbol{\phi}_{ij} = \boldsymbol{\mu}_j, \quad i \in V, j \in \mathcal{B}_i, \\ a_i = \psi_{ij}, & \psi_{ij} = a_j, \quad i \in V, j \in \mathcal{B}_i \end{array} \tag{5}$$

where $F(\boldsymbol{\Theta}_i) = \sum_{n=1}^{N_i} \log p(\mathbf{x}_{in} | \mathbf{W}_i, \boldsymbol{\mu}_i, a_i^{-1})$. The augmented Lagrangian is

$$\mathcal{L}(\boldsymbol{\Phi}_i) = -F(\boldsymbol{\Theta}_i) +$$

$$\sum_{i \in V} \sum_{j \in \mathcal{B}_i} \left( \boldsymbol{\lambda}_{ij1}^{\mathrm{T}}(\mathbf{W}_i - \boldsymbol{\rho}_{ij}) + \boldsymbol{\lambda}_{ij2}^{\mathrm{T}}(\boldsymbol{\rho}_{ij} - \mathbf{W}_j) \right) + \sum_{i \in V} \sum_{j \in \mathcal{B}_i} \left( \boldsymbol{\gamma}_{ij1}^{\mathrm{T}}(\boldsymbol{\mu}_i - \boldsymbol{\phi}_{ij}) + \boldsymbol{\gamma}_{ij2}^{\mathrm{T}}(\boldsymbol{\phi}_{ij} - \boldsymbol{\mu}_j) \right)$$

$$+ \sum_{i \in V} \sum_{j \in \mathcal{B}_i} \left( \beta_{ij1}(a_i - \psi_{ij}) + \beta_{ij2}(\psi_{ij} - a_j) \right) + \frac{\eta}{2} \sum_{i \in V} \sum_{j \in \mathcal{B}_i} (\|\mathbf{W}_i - \boldsymbol{\rho}_{ij}\|^2 + \|\boldsymbol{\rho}_{ij} - \mathbf{W}_j\|^2)$$

$$+ \frac{\eta}{2} \sum_{i \in V} \sum_{j \in \mathcal{B}_i} (\|\boldsymbol{\mu}_i - \boldsymbol{\phi}_{ij}\|^2 + \|\boldsymbol{\phi}_{ij} - \boldsymbol{\mu}_j\|^2) + \frac{\eta}{2} \sum_{i \in V} \sum_{j \in \mathcal{B}_i} ((a_i - \psi_{ij})^2 + (\psi_{ij} - a_j)^2) \tag{6}$$

where $\boldsymbol{\Phi}_i = \{\mathbf{W}_i, \boldsymbol{\mu}_i, a_i, \boldsymbol{\rho}_{ij}, \boldsymbol{\phi}_{ij}, \psi_{ij}; i \in V, j \in \mathcal{B}_i\}$ and $\{\boldsymbol{\lambda}_{ijk}\}, \{\boldsymbol{\gamma}_{ijk}\}, \{\beta_{ijk}\}$ with $k = 1, 2$ are the Lagrange multipliers. The scalar value $\eta$ gives us control over the convergence speed of the algorithm. With reasonably large positive $\eta$, the overall optimization converges fairly quickly [10]. We will explore the converging behaviour with respect to various $\eta$ in synthetic data experiments.

Just like in a standard EM approach, we minimize the upper bound of $\mathcal{L}(\boldsymbol{\Phi}_i)$. Exploiting the posterior density in (4), we compute the expected mean and variance of latent variables in each node as

$$\mathbb{E}[\mathbf{z}_{in}] = \mathbf{L}_i^{-1} \mathbf{W}_i^{\mathrm{T}} (\mathbf{x}_{in} - \boldsymbol{\mu}_i), \qquad \mathbb{E}[\mathbf{z}_{in} \mathbf{z}_{in}^{\mathrm{T}}] = a_i^{-1} \mathbf{L}_i^{-1} + \mathbb{E}[\mathbf{z}_{in}] \mathbb{E}[\mathbf{z}_{in}]^{\mathrm{T}}. \tag{7}$$

Maximization of the completed likelihood Lagrangian derived from (6) yields

$$\mathbf{W}_i^{(t+1)} = \left\{ a_i \sum_{n=1}^{N_i} (\mathbf{x}_{in} - \boldsymbol{\mu}_i) \mathbb{E}[\mathbf{z}_{in}]^{\mathrm{T}} - 2\boldsymbol{\lambda}_i^{(t)} + \eta \sum_{j \in B_i} \left( \mathbf{W}_i^{(t)} + \mathbf{W}_j^{(t)} \right) \right\} \cdot \left( a_i \sum_{n=1}^{N_i} \mathbb{E}[\mathbf{z}_{in} \mathbf{z}_{in}^{\mathrm{T}}] + 2\eta |\mathcal{B}_i| \mathbf{I} \right)^{-1}, \tag{8}$$

$$\boldsymbol{\mu}_i^{(t+1)} = \left\{ a_i \sum_{n=1}^{N_i} \left( \mathbf{x}_{in} - \mathbf{W}_i \mathbb{E}[\mathbf{z}_{in}] \right) - 2\boldsymbol{\gamma}_i^{(t)} + \eta \sum_{j \in B_i} \left( \boldsymbol{\mu}_i^{(t)} + \boldsymbol{\mu}_j^{(t)} \right) \right\} \cdot (N_i a_i + 2\eta |\mathcal{B}_i|)^{-1}, \tag{9}$$

$$\boldsymbol{\lambda}_i^{(t+1)} = \boldsymbol{\lambda}_i^{(t)} + \frac{\eta}{2} \sum_{j \in \mathcal{B}_i} \left\{ \mathbf{W}_i^{(t+1)} - \mathbf{W}_j^{(t+1)} \right\}, \tag{10}$$

$$\boldsymbol{\gamma}_i^{(t+1)} = \boldsymbol{\gamma}_i^{(t)} + \frac{\eta}{2} \sum_{j \in \mathcal{B}_i} \left\{ \boldsymbol{\mu}_i^{(t+1)} - \boldsymbol{\mu}_j^{(t+1)} \right\}, \tag{11}$$

$$\beta_i^{(t+1)} = \beta_i^{(t)} + \frac{\eta}{2} \sum_{j \in \mathcal{B}_i} \left\{ a_i^{(t+1)} - a_j^{(t+1)} \right\}. \tag{12}$$

For $a_i$, we solve the quadratic equation

$$0 = -\frac{N_i D}{2} + 2\eta |\mathcal{B}_i| a_i^{(t+1)^2} + a_i^{(t+1)} \cdot \left\{ 2\beta_i^{(t)} - \eta \sum_{j \in B_i} \left( a_i^{(t)} + a_j^{(t)} \right) - \sum_{n=1}^{N_i} \mathbb{E}[\mathbf{z}_{in}]^{\mathrm{T}} \mathbf{W}_i^{\mathrm{T}} (\mathbf{x}_{in} - \boldsymbol{\mu}_i) \right.$$

$$+ \frac{1}{2} \sum_{n=1}^{N_i} \left\{ \|\mathbf{x}_{in} - \boldsymbol{\mu}_i\|^2 + tr \left[ \mathbb{E}[\mathbf{z}_{in} \mathbf{z}_{in}^{\mathrm{T}}] \mathbf{W}_i^{\mathrm{T}} \mathbf{W}_i \right] \right\} \Bigg\}. \tag{13}$$

The overall distributed EM algorithm for D-PPCA is summarized in Algorithm 1. Detailed derivation can be found in the supplementary material.

**Algorithm 1** Distributed Probabilistic PCA (D-PPCA)

---

**Require:** For every node $i$ initialize $\mathbf{W}_i^{(0)}, \boldsymbol{\mu}_i^{(0)}, a_i^{(0)}$ randomly and set $\boldsymbol{\lambda}_i^{(0)} = \mathbf{0}, \boldsymbol{\gamma}_i^{(0)} = \mathbf{0}, \beta_i^{(0)} = 0$.
  **for** $t = 0, 1, 2, ...$ until convergence **do**
    **for all** $i \in V$ **do**
      [E-step] Compute $\mathbb{E}[\mathbf{z}_{in}]$ and $\mathbb{E}[\mathbf{z}_{in}\mathbf{z}_{in}^{\mathrm{T}}]$ via (7).
      [M-step] Compute $\mathbf{W}_i^{(t+1)}, \boldsymbol{\mu}_i^{(t+1)}, a_i^{(t+1)}$ via (8,9,13).
    **end for**
    **for all** $i \in V$ **do**
      Broadcast $\mathbf{W}_i^{(t+1)}, \boldsymbol{\mu}_i^{(t+1)}$, and $a_i^{(t+1)}$ to all neighbors of $i \in \mathcal{B}_i$.
    **end for**
    **for all** $i \in V$ **do**
      Compute $\boldsymbol{\lambda}_i^{(t+1)}, \boldsymbol{\gamma}_i^{(t+1)}$, and $\beta_i^{(t+1)}$ via (10-12).
    **end for**
  **end for**

---

## 3.2 Missing Data D-PPCA

Traditional PPCA is an effective tool for dealing with data missing-at-random (MAR) in traditional PCA [18]. While more sophisticated methods including variational approximations, c.f., [18] are possible direct use of PPCA is often sufficient in practice. Hence, we adopt D-PPCA as a method to deal with missing data in a distributed consensus setting.

Generalization to missing data D-PPCA from D-PPCA is straightforward and follows [18]. From the perspective of ADMM-based learning the only modifications comes in the form of adjusted terms for local data summaries. For instance, in (9) the data summary term $\sum_{n=1}^{N_i}(\mathbf{x}_{in} - \mathbf{W}_i\mathbb{E}[\mathbf{z}_{in}])$ becomes

$$\sum_{n \in O_{i,f}} x_{i,n,f} - \mathbf{w}_{i,f}^T \mathbb{E}[\mathbf{z}_{in}], \tag{14}$$

where $f = 1, \ldots, D$ is the index of feature, $O_{i,f}$ is the set of samples in node $i$ that have the feature $f$ present, $x_{i,n,f}$ is the value of the present feature, and $\mathbf{w}_{i,f}^T$ is the $f$-th row of matrix $\mathbf{W}_i$. Similar expressions can be derived for other local parameters. Note that (10-12) incur no changes.

# 4 D-PPCA for Structure from Motion (SfM)

In this section, we consider a specific formulation of the modified distributed probabilistic PCA for application in affine SfM. In SfM, our goal is to estimate the 3D location of $N$ points on a rigid object based on corresponding 2-D points observed from multiple cameras (or views). The dimension $D$ of our measurement matrix is thus twice the number of frames each camera observed. A simple and effective way to solve this problem is the factorization method [19]. Given a 2D (image coordinate) measurement matrix $\mathbf{X}$, of size $2 \cdot \#frames \times \#points$, the matrix is factorized into a $2 \cdot \#frames \times 3$ motion matrix $\mathbf{M}$ and the $3 \times \#points$ 3D structure matrix $\mathbf{S}$. In the centralized setting this can be easily computed using SVD on $\mathbf{X}$. Equivalently, the estimates of $\mathbf{M}$ and $\mathbf{S}$ can be found using inference and learning in a centralized PPCA, where $\mathbf{M}$ is treated as the PPCA parameter and $\mathbf{S}$ is the latent structure. There we obtain additional estimates of the variance of structure $\mathbf{S}$, which are not immediately available from the factorization approach (although, they can be found).

However, the above defined ($2 \cdot \#frames \times \#points$) data structure of $\mathbf{X}$ is not amenable to distribution of different views (cameras, nodes), as considered in Section 3 of D-PPCA. Namely, D-PPCA assumes that the distribution is accomplished by splitting the data matrix $\mathbf{X}$ into sets of non-overlapping columns, one for each node. Here, however, we seek to distribute the rows of matrix $\mathbf{X}$, i.e., a set of (subsequent) frames is to be assigned to each node/camera.

Hence, to apply the D-PPCA framework to SfM we need to swap the role of rows and columns, i.e., consider modeling of $\mathbf{X}^{\mathrm{T}}$. This, subsequently, means that the 3D scene structure (which is to be shared across all nodes in the network) will be treated as the D-PPCA *parameter*. The latent D-PPCA variables will model the unknown and uncertain motion of each camera (and/or object in its view).

Specifically, we will consider the model

$$\mathbf{X}_i^{\mathrm{T}} = \mathbf{W} \cdot \mathbf{Z}_i + \mathbf{E}_i \tag{15}$$

where $\mathbf{X}_i^{\mathrm{T}}$ is the matrix of image coordinates of all points in node (camera) $i$ of size $\#points \times 2 \cdot \#frames$ in node $i$, $\mathbf{W}$ is the $\#points \times 3$ 3D structure (D-PPCA parameter) matrix and $\mathbf{Z}_i$ is the $3 \times 2 \cdot \#frames$ motion matrix of node $i$.

One should note that we have implicitly assumed, in a standard D-PPCA manner, that each column of $\mathbf{Z}_i$ is iid and distributed as $\mathcal{N}(0, \mathbf{I})$. However, each pair of subsequent $\mathbf{Z}_i$ columns represents one $3 \times 2$ affine motion matrix. While those columns are not truly independent our experiments (as demonstrated in Section 5) show that this assumption is not detrimental in practice. Remaining task is simply following the same process we did to derive D-PPCA.

Missing data in SfM will be handled using the formalism presented in Sec. 3.2. Strictly speaking, the model of data missing-at-random is not always applicable to SfM. The reason is that occlusions, the main source of missing data, cannot be treated as a random process. Instead, this setting corresponds to data missing-not-at-random [18] (MNAR). If treated blindly, this may introduce bias in the estimated models. However, as we demonstrate in experiments this assumption does not adversely affect SfM when the number of missing points is within a reasonable range.

## 5 Experiments

In our experiments we first study the general convergence properties of the D-PPCA algorithm in a controlled synthetic setting. We then apply the D-PPCA to a set of SfM problems, both on synthetic and on real data.

### 5.1 Empirical Convergence Analysis

Using synthetic data generated from Gaussian distribution, we observed that D-PPCA works well regardless of the number of network nodes, topology, choice of the parameter $\eta$ or even with missing values in both MAR and MNAR cases. Detailed results for the syntehtic data is provided in the supplementary materials.

### 5.2 Affine Structure from Motion

We now show that the modified D-PPCA can be used as an effective framework for distributed affine SfM. We first show results in a controlled environment with synthetic data and then report results on data from real video sequences. We assume that correspondences across frames and cameras are known. For the missing values of MNAR case, we either used the actual occlusions to induce missing points or simulated consistently missing points over several frames.

#### 5.2.1 Synthetic Data (Cube)

We first generated synthetic data with a rotating unit cube and 5 cameras facing the cube in a 3D space, similar to synthetic experiments in [6]. The cube is centered at the origin of the space and rotates $30°$ counterclockwise. We extracted 8 cube points projected on each camera view every $6°$, i.e. each camera observed 5 frames. Cameras are placed on a skewed plane, making elevation along $z$-axis as shown in Fig. 2a. For all synthetic and real SfM experiments, we picked $\eta = 10$ and initialized $\mathbf{W}_i$ matrix with feature point coordinates of the first frame visible in the $i$-th camera with some small noise. The convergence criterion for D-PPCA for SfM was set as $10^{-3}$ relative error. To measure the performance, we computed maximum subspace angle between the ground truth 3D coordinates and our estimated 3D structure matrix. For comparison, we conducted traditional SVD-based SfM on the same data. In noise free case, D-PPCA for SfM always yielded the same performance as SVD-based SfM with near $0°$.

We also tested D-PPCA for SfM with noisy and missing-value cases. First, we generated 20 independent samples of all 25 frames with 10 different noise levels. Then we ran D-PPCA 20 times on each of the independent sample and averaged the final structure estimates. As Fig. 2b shows, we found that D-PPCA for SfM is fairly robust to noise and tends to stabilize even as the noise level

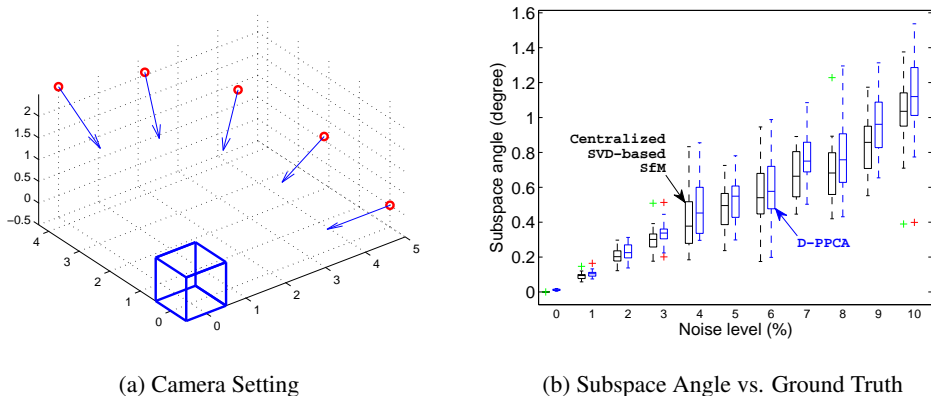

|                         |                            |
|:-----------------------:|:--------------------------:|
| (a) Camera Setting      | (b) Subspace Angle vs. Ground Truth |

Figure 2: Rotating unit cube with multiple cameras. Red circles are camera locations and blue arrows indicate each camera's facing direction. Green and red crosses in the right plot are outliers for centralized SVD-based SfM and D-PPCA for SfM, respectively.

increases. The mean subspace angle tends to be slightly larger than that estimated by the centralized SVD SfM, however both reside within the overlapping confidence intervals. Considering MAR missing values, we obtained $1.66°$ for 20% missing points averaged over 10 different missing point samples. In MNAR case with actual occlusions considered, D-PPCA yielded, relatively larger, $20°$ error. Intuitively, this is because the missing points in the scene are naturally not random. However, we argue that D-PPCA can still handle missing points given the evidence below.

### 5.2.2   Real Data

For real data experiement, we first applied D-PPCA for SfM on the Caltech 3D Objects on Turntable dataset [20]. The dataset provides various objects rotating on a turntable under different lighting conditions. The views of most objects were taken every $5°$ which make it challenging to extract feature points with correspondence across frames. Instead, we used a subset of the dataset which provides views taken every degree. This subset contains images of 5 objects. To simulate multiple cameras, we adopted a setting similar to that of [6]. We first extracted first $30°$ images of each object. We then used KLT [21] implementation in Voodoo Camera Tracker[1] to extract feature points with correspondence. Lastly, we sequentially and equally partitioned the 30 images into 5 nodes to simulate 5 cameras. Thus, each camera observes 6 frames. Table 1 shows the 5 objects and statistics of feature points we extracted from the objects. We used $\eta = 10$ and convergence criterion $10^{-3}$. Due to the lack of the ground truth 3D coordinates, we compared the subspace angles between the structure inferred using the traditional centralized SVD-based SfM and the D-PPCA-based SfM. Results are shown in Table 1 as the mean and variance of 20 independent runs. 10% MAR and MNAR results are also provided in the table.

Experimenal results indicate existance of differences between the reconstructions obtained by centralized factorization approach and that of D-PPCA. However, the differences are small, depend on the object in question, and almost always include, within their confidence, the factorization result. Qualitative examination reveals no noticable differences. Moreover, re-projecting back to the camera coordinate space resulted in close matching with the tracked feature points, as shown in videos provided in supplementary materials.

We also tested the utility of D-PPCA for SfM on the Hopkins155 dataset [22]. We adopted virtually identical experimental setting as in [6]. We collected 135 single-object sequences containing image coordinates of points and we simulated multi-camera setting by partitioning the frames sequentially and almost equally for 5 nodes and the network was connected using ring topology. Again, we computed maximum subspace angle between centralized SVD-based SfM and distributed D-PPCA for SfM. We chose the convergence criterion as $10^{-3}$. Average maximum subspace angle between

Table 1: Caltech 3D Objects on Turntable dataset statistics and quantitative results. Green dots indicate feature points tracked with correspondance across all 30 frames. All results ran 20 independent initializations. MAR results provide variances over both various initializations and missing value settings.

| Object | BallSander | BoxStuff | Rooster | Standing | StorageBin |
|---|---|---|---|---|---|
| |  |  |  |  | 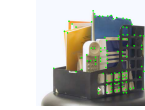 |
| # Points | 62 | 67 | 189 | 310 | 102 |
| # Frames | 30 | 30 | 30 | 30 | 30 |
| Subspace angle b/w centralized SVD SfM and D-PPCA (degree) | | | | | |
| Mean | 1.4848 | 1.4397 | 1.4767 | 2.6221 | 0.4463 |
| Variance | 0.4159 | 0.4567 | 0.9448 | 1.6924 | 1.2002 |
| Subspace angle b/w fully observable centralized PPCA SfM and D-PPCA with MAR (degree) | | | | | |
| Mean | 6.2991 | 2.1556 | 5.2506 | 7.6492 | 2.8358 |
| Var.(init) | 4.3562 | 0.1351 | 3.8810 | 6.6424 | 1.3591 |
| Var.(miss) | 0.5729 | 0.0161 | 0.1755 | 0.7603 | 0.0444 |
| Subspace angle b/w fully observable centralized PPCA SfM and D-PPCA with MNAR (degree) | | | | | |
| Mean | 3.1405 | 6.4664 | 5.8027 | 9.2661 | 3.7965 |
| Variance | 0.0124 | 3.1955 | 2.4333 | 2.9720 | 0.0089 |

D-PPCA for SfM and SVD-based SfM for all objects was $3.97°$ with variance 7.06. However, looking into the result more carefully, we found that even with substantially larger subspace angle, 3D structure estimates were similar to that of SVD-based SfM only with orthogonal ambiguity issue. Moreover, more than 53% of all objects yielded the subspace angle below $1°$, 77% of them below $5°$ and more than 94% were less than $15°$. With 10% MAR, we obtained the mean $20.07°$ with variance $27.94°$ with about 18% of them below $1°$, 56% of them below $5°$ and more than 70% of them less than the mean. We could not perform MNAR experiments on Hopkins as the ground truth occlusion information is not provided with the dataset.

## 6 Discussion and Future Work

In this work we introduced a general approach for learning parameters of traditional centralized probabilistic models, such as PPCA, in a distributed setting. Our synthetic data experiments showed that the proposed algorithm is robust to choices of initial parameters and, more importantly, is not adversely affected by variations in network size, topology or missing values. In the SfM problems, the algorithm can be effectively used to distribute computation of 3D structure and motion in camera networks, while retaining the probabilistic nature of the original model.

Despite its promising performance D-PPCA for SfM exhibits some limitations. In particular, we assume the independence of the affine motion matrix parameters in (15). The assumption is clearly inconsistent with the modeling of motion on the SE(3) manifold. However, our experiments demonstrate that, in practice, this violation is not crucial. This shortcoming can be amended in one of several possible ways. One can reduce the iid assumption of individual samples to that of subsequent columns (i.e., full 3x2 motion matrices). Our additional experiments, not reported here, indicate no discernable utility of this approach. A more principled approach would be to define priors for motion matrices compatible with SE(3), using e.g., [23]. While appealing, the priors would render the overall model non-linear and would require additional algorithmic considerations, perhaps in the spirit of [1].

### Acknowledgments

This work was supported in part by the National Science Foundation under Grant No. IIS 0916812.

## Footnotes

[1] http://www.digilab.uni-hannover.de/docs/manual.html

# References

[1] Roberto Tron and Rene Vidal. Distributed Computer Vision Algorithms. *IEEE Signal Processing Magazine*, 28:32–45, 2011.

[2] A.Y. Yang, S. Maji, C.M. Christoudias, T. Darrell, J. Malik, and S.S. Sastry. Multiple-view Object Recognition in Band-limited Distributed Camera Networks. In *Distributed Smart Cameras, 2009. ICDSC 2009. Third ACM/IEEE International Conference on*, 30 2009-sept. 2 2009.

[3] Richard J. Radke. A Survey of Distributed Computer Vision Algorithms. In Hideyuki Nakashima, Hamid Aghajan, and Juan Carlos Augusto, editors, *Handbook of Ambient Intelligence and Smart Environments*. Springer Science+Business Media, LLC, 2010.

[4] A. Wiesel and A.O. Hero. Decomposable Principal Component Analysis. *Signal Processing, IEEE Transactions on*, 57(11):4369–4377, 2009.

[5] Sergio V. Macua, Pavle Belanovic, and Santiago Zazo. Consensus-based Distributed Principal Component Analysis in Wireless Sensor Networks. In *Signal Processing Advances in Wireless Communications (SPAWC), 2010 IEEE Eleventh International Workshop on*, pages 1–5, June 2010.

[6] Roberto Tron and Rene Vidal. Distributed Computer Vision Algorithms Through Distributed Averaging. In *IEEE Conference on Computer Vision and Pattern Recognition*, pages 57–63, 2011.

[7] Lin Xiao, Stephen Boyd, and Sanjay Lall. A Scheme for Robust Distributed Sensor Fusion Based on Average Consensus. In *International Conference on Information Processing in Sensor Networks*, pages 63–70, April 2005.

[8] R. Olfati-Saber. Distributed Kalman Filtering for Sensor Networks. In *Decision and Control, 2007 46th IEEE Conference on*, pages 5492 –5498, dec. 2007.

[9] Bi Song, A.T. Kamal, C. Soto, Chong Ding, J.A. Farrell, and A.K. Roy-Chowdhury. Tracking and Activity Recognition Through Consensus in Distributed Camera Networks. *Image Processing, IEEE Transactions on*, 19(10):2564 –2579, oct. 2010.

[10] P.A. Forero, A. Cano, and G.B. Giannakis. Distributed Clustering Using Wireless Sensor Networks. *Selected Topics in Signal Processing, IEEE Journal of*, 5(4):707 –724, aug. 2011.

[11] Sam Roweis and Zoubin Ghahramani. A Unifying Review of Linear Gaussian Models. *Neural Computation*, 11:305–345, 1999.

[12] Ami Wiesel, Yonina C. Eldar, and Alfred O. Hero. Covariance Estimation in Decomposable Gaussian Graphical Models. *IEEE Transactions on Signal Processing*, 58(3):1482–1492, 2010.

[13] Andrew R. Conn, Nicholas I. M. Gould, and Philippe L. Toint. A globally convergent augmented Lagrangian algorithm for optimization with general constraints and simple bounds. *SIAM J. Numer. Anal.*, 28:545–572, February 1991.

[14] Robert Michael Lewis and Virginia Torczon. A Globally Convergent Augmented Lagrangian Pattern Search Algorithm for Optimization with General Constraints and Simple Bounds. *SIAM J. on Optimization*, 12:1075–1089, April 2002.

[15] Stephen Boyd, Neal Parikh, Eric Chu, Borja Peleato, and Jonathan Eckstein. Distributed Optimization and Statistical Learning via the Alternating Direction Method of Multipliers. In Michael Jordan, editor, *Foundations and Trends in Machine Learning*, volume 3, pages 1–122. Now Publishers, 2011.

[16] Pedro A. Forero, Alfonso Cano, and Geogios B. Giannakis. Consensus-Based Distributed Support Vector Machines. *Journal of Machine Learning Research*, 11:1663–1707, 2010.

[17] Michael E. Tipping and Chris M. Bishop. Probabilistic Principal Component Analysis. *Journal of the Royal Statistical Society, Series B*, 61:611–622, 1999.

[18] Alexander Ilin and Tapani Raiko. Practical Approaches to Principal Component Analysis in the Presence of Missing Values. *Journal of Machine Learning Research*, 11:1957–2000, 2010.

[19] Carlo Tomasi and Takeo Kanade. Shape and motion from image streams under orthography: a factorization method. *International Journal of Computer Vision*, 9:137–154, 1992. 10.1007/BF00129684.

[20] Pierre Moreels and Pietro Perona. Evaluation of Features Detectors and Descriptors based on 3D Objects. *International Journal of Computer Vision*, 73(3):263–284, July 2007.

[21] Carlo Tomasi and Takeo Kanade. Detection and Tracking of Point Features. Technical Report CMU-CS-91-132, Carnegie Mellon University, April 1991.

[22] Roberto Tron and Rene Vidal. A Benchmark for the Comparison of 3-D Motion Segmentation Algorithms. In *IEEE International Conference on Computer Vision and Pattern Recognition*, 2007.

[23] Yasuko Chikuse. *Statistics on Special Manifolds*, volume 174 of *Lecture Notes in Statistics*. Springer, 1 edition, February 2003.

